# On the Convergence of Eigenspaces in Kernel Principal Component Analysis

**Laurent Zwald**
Département de Mathématiques,
Université Paris-Sud,
Bât. 425, F-91405 Orsay, France
Laurent.Zwald@math.u-psud.fr

**Gilles Blanchard**
Fraunhofer First (IDA),
Kékuléstr. 7, D-12489 Berlin, Germany
blanchar@first.fhg.de

## Abstract

This paper presents a non-asymptotic statistical analysis of Kernel-PCA with a focus different from the one proposed in previous work on this topic. Here instead of considering the reconstruction error of KPCA we are interested in approximation error bounds for the eigenspaces themselves. We prove an upper bound depending on the spacing between eigenvalues but not on the dimensionality of the eigenspace. As a consequence this allows to infer stability results for these estimated spaces.

## 1 Introduction.

Principal Component Analysis (PCA for short in the sequel) is a widely used tool for data dimensionality reduction. It consists in finding the most relevant lower-dimension projection of some data in the sense that the projection should keep as much of the variance of the original data as possible. If the target dimensionality of the projected data is fixed in advance, say $D$ – an assumption that we will make throughout the present paper – the solution of this problem is obtained by considering the projection on the span $S_D$ of the first $D$ eigenvectors of the covariance matrix. Here by 'first $D$ eigenvectors' we mean eigenvectors associated to the $D$ largest eigenvalues counted with multiplicity; hereafter with some abuse the span of the first $D$ eigenvectors will be called "$D$-eigenspace" for short when there is no risk of confusion.

The introduction of the 'Kernel trick' has allowed to extend this methodology to data mapped in a kernel feature space, then called KPCA [8]. The interest of this extension is that, while still linear in feature space, it gives rise to *nonlinear* interpretation in original space – vectors in the kernel feature space can be interpreted as nonlinear functions on the original space.

For PCA as well as KPCA, the true covariance matrix (resp. covariance operator) is not known and has to be estimated from the available data, an procedure which in the case of Kernel spaces is linked to the so-called Nyström approximation [13]. The subspace given as an output is then obtained as $D$-eigenspace $\widehat{S}_D$ of the *empirical* covariance matrix or operator. An interesting question from a statistical or learning theoretical point of view is then, how reliable this estimate is.

This question has already been studied [10, 2] from the point of view of the *reconstruction*

*error* of the estimated subspace. What this means is that (assuming the data is centered in Kernel space for simplicity) the average reconstruction error (square norm of the distance to the projection) of $\widehat{S}_D$ converges to the (optimal) reconstruction error of $S_D$ and that bounds are known about the rate of convergence. However, this does not tell us much about the convergence of $S_D$ to $\widehat{S}_D$ – since two very different subspaces can have a very similar reconstruction error, in particular when some eigenvalues are very close to each other (the gap between the eigenvalues will actually appear as a central point of the analysis to come).

In the present work, we set to study the behavior of these $D$-eigenspaces themselves: we provide finite sample bounds describing the closeness of the $D$-eigenspaces of the empirical covariance operator to the true one. There are several broad motivations for this analysis. First, the reconstruction error alone is a valid criterion only if one really plans to perform dimensionality reduction of the data and stop there. However, PCA is often used merely as a *preprocessing* step and the projected data is then submitted to further processing (which could be classification, regression or something else). In particular for KPCA, the projection subspace in the kernel space can be interpreted as a subspace of *functions* on the original space; one then expects these functions to be relevant for the data at hand and for some further task (see e.g. [3]). In these cases, if we want to analyze the full procedure (from a learning theoretical sense), it is desirable to have a more precise information on the selected subspace than just its reconstruction error. In particular, from a learning complexity point of view, it is important to ensure that functions used for learning stay in a set of limited complexity, which is ensured if the selected subspace is stable (which is a consequence of its convergence).

The approach we use here is based on perturbation bounds and we essentially walk in the steps pioneered by Kolchinskii and Giné [7] (see also [4]) using tools of operator perturbation theory [5]. Similar methods have been used to prove consistency of spectral clustering [12, 11]. An important difference here is that we want to study directly the convergence of the whole subspace spanned by the first $D$ eigenvectors instead of the separate convergence of the individual eigenvectors; in particular we are interested in how $D$ acts as a complexity parameter. The important point in our main result is that it does not: only the gap between the $D$-th and the $(D + 1)$-th eigenvalue comes into account. This means that there in no increase in complexity (as far as this bound is concerned: of course we cannot exclude that better bounds can be obtained in the future) between estimating the $D$-th eigenvector alone or the span of the first $D$ eigenvectors.

Our contribution in the present work is thus

- to adapt the operator perturbation result of [7] to $D$-eigenspaces.
- to get non-asymptotic bounds on the approximation error of Kernel-PCA eigenspaces thanks to the previous tool.

In section 2 we introduce shortly the notation, explain the main ingredients used and obtain a first bound based on controlling separately the first $D$ eigenvectors, and depending on the dimension $D$. In section 3 we explain why the first bound is actually suboptimal and derive an improved bound as a consequence of an operator perturbation result that is more adapted to our needs and deals directly with the $D$-eigenspace as a whole. Section 4 concludes and discusses the obtained results. Mathematical proofs are found in the appendix.

## 2 First result.

**Notation.** The interest variable $X$ takes its values in some measurable space $\mathcal{X}$, following the distribution $P$. We consider KPCA and are therefore primarily interested in the mapping of $X$ into a reproducing kernel Hilbert space $\mathcal{H}$ with kernel function $k$ through the

feature mapping $\varphi(x) = k(x, \cdot)$. The objective of the kernel PCA procedure is to recover a $D$-dimensional subspace $S_D$ of $\mathcal{H}$ such that the projection of $\varphi(X)$ on $S_D$ has maximum averaged squared norm.

All operators considered in what follows are Hilbert-Schmidt and the norm considered for these operators will be the Hilbert-Schmidt norm unless precised otherwise. Furthermore we only consider symmetric nonnegative operators, so that they can be diagonalized and have a discrete spectrum.

Let $C$ denote the covariance operator of variable $\varphi(X)$. To simplify notation we assume that nonzero eigenvalues $\lambda_1 > \lambda_2 > \ldots$ of $C$ are all simple (This is for convenience only. In the conclusion we discuss what changes have to be made if this is not the case). Let $\phi_1, \phi_2, \ldots$ be the associated eigenvectors. It is well-known that the optimal $D$-dimensional reconstruction space is $S_D = \mathrm{span}\{\phi_1, \ldots, \phi_D\}$. The KPCA procedure approximates this objective by considering the empirical covariance operator, denoted $C_n$, and the subspace $\widehat{S}_D$ spanned by its first $D$ eigenvectors. We denote $P_{S_D}, P_{\widehat{S}_D}$ the orthogonal projectors on these spaces.

**A first bound.**   Broadly speaking, the main steps required to obtain the type of result we are interested in are

1. A non-asympotic bound on the (Hilbert-Schmidt) norm of the difference between the empirical and the true covariance operators;

2. An operator perturbation result bounding the difference between spectral projectors of two operators by the norm of their difference.

The combination of these two steps leads to our goal. The first step consists in the following Lemma coming from [9]:

**Lemma 1 (Corollary 5 of [9])** *Supposing that* $\sup_{x \in \mathcal{X}} k(x, x) \leq M$, *with probability greater than* $1 - e^{-\xi}$,

$$\|C_n - C\| \leq \frac{2M}{\sqrt{n}} \left(1 + \sqrt{\frac{\xi}{2}}\right).$$

As for the second step, [7] provides the following perturbation bound (see also e.g. [12]):

**Theorem 2 (Simplified Version of [7], Theorem 5.2 )** *Let $A$ be a symmetric positive Hilbert-Schmidt operator of the Hilbert space $\mathcal{H}$ with simple positive eigenvalues $\lambda_1 > \lambda_2 > \ldots$ For an integer $r$ such that $\lambda_r > 0$, let $\widetilde{\delta}_r = \delta_r \wedge \delta_{r-1}$ where $\delta_r = \frac{1}{2}(\lambda_r - \lambda_{r+1})$. Let $B \in HS(\mathcal{H})$ be another symmetric operator such that $\|B\| < \widetilde{\delta}_r/2$ and $(A + B)$ is still a positive operator with simple nonzero eigenvalues.*

*Let $P_r(A)$ (resp. $P_r(A + B)$) denote the orthogonal projector onto the subspace spanned by the $r$-th eigenvector of $A$ (resp. $(A + B)$). Then, these projectors satisfy:*

$$\|P_r(A) - P_r(A + B)\| \leq \frac{2\|B\|}{\widetilde{\delta}_r}.$$

**Remark about the Approximation Error of the Eigenvectors:**   let us recall that a control over the Hilbert-Schmidt norm of the projections onto eigenspaces imply a control on the approximation errors of the eigenvectors themselves. Indeed, let $\phi_r, \psi_r$ denote the (normalized) $r$-th eigenvectors of the operators above with signs chosen so that $\langle \phi_r, \psi_r \rangle > 0$. Then

$$\|P_{\phi_r} - P_{\psi_r}\|^2 = 2(1 - \langle \phi_r, \psi_r \rangle^2) \geq 2(1 - \langle \phi_r, \psi_r \rangle) = \|\phi_r - \psi_r\|^2.$$

Now, the orthogonal projector on the direct sum of the first $D$ eigenspaces is the sum $\sum_{r=1}^{D} P_r$. Using the triangle inequality, and combining Lemma 1 and Theorem 2, we conclude that with probability at least $1 - e^{-\xi}$ the following holds:

$$\left\| P_{S_D} - P_{\widehat{S}_D} \right\| \leq \left( \sum_{r=1}^{D} \widetilde{\delta}_r^{-1} \right) \frac{4M}{\sqrt{n}} \left( 1 + \sqrt{\frac{\xi}{2}} \right),$$

provided that $n \geq 16M^2 \left( 1 + \sqrt{\frac{\xi}{2}} \right)^2 (\sup_{1 \leq r \leq D} \widetilde{\delta}_r^{-2})$. The disadvantage of this bound is that we are penalized on the one hand by the (inverse) gaps between the eigenvalues, and on the other by the dimension $D$ (because we have to sum the inverse gaps from 1 to D). In the next section we improve the operator perturbation bound to get an improved result where only the gap $\delta_D$ enters into account.

## 3   Improved Result.

We first prove the following variant on the operator perturbation property which better corresponds to our needs by taking directly into account the projection on the first $D$ eigenvectors at once. The proof uses the same kind of techniques as in [7].

**Theorem 3** *Let $A$ be a symmetric positive Hilbert-Schmidt operator of the Hilbert space $\mathcal{H}$ with simple nonzero eigenvalues $\lambda_1 > \lambda_2 > \ldots$ Let $D > 0$ be an integer such that $\lambda_D > 0$, $\delta_D = \frac{1}{2}(\lambda_D - \lambda_{D+1})$. Let $B \in HS(\mathcal{H})$ be another symmetric operator such that $\|B\| < \delta_D/2$ and $(A + B)$ is still a positive operator. Let $P^D(A)$ (resp. $P^D(A + B)$) denote the orthogonal projector onto the subspace spanned by the first $D$ eigenvectors $A$ (resp. $(A + B)$). Then these satisfy:*

$$\|P^D(A) - P^D(A + B)\| \leq \frac{\|B\|}{\delta_D}. \tag{1}$$

This then gives rise to our main result on KPCA:

**Theorem 4** *Assume that $\sup_{x \in \mathcal{X}} k(x, x) \leq M$. Let $S_D, \widehat{S}_D$ be the subspaces spanned by the first $D$ eigenvectors of $C$, resp. $C_n$ defined earlier. Denoting $\lambda_1 > \lambda_2 > \ldots$ the eigenvalues of $C$, if $D > 0$ is such that $\lambda_D > 0$, put $\delta_D = \frac{1}{2}(\lambda_D - \lambda_{D+1})$ and*

$$B_D = \frac{2M}{\delta_D} \left( 1 + \sqrt{\frac{\xi}{2}} \right).$$

*Then provided that $n \geq B_D^2$, the following bound holds with probability at least $1 - e^{-\xi}$:*

$$\left\| P_{S_D} - P_{\widehat{S}_D} \right\| \leq \frac{B_D}{\sqrt{n}}. \tag{2}$$

*This entails in particular*

$$\widehat{S}_D \subset \left\{ g + h, g \in S_D, h \in S_D^{\perp}, \|h\|_{\mathcal{H}_k} \leq 2B_D \, n^{-\frac{1}{2}} \|g\|_{\mathcal{H}_k} \right\}. \tag{3}$$

The important point here is that the approximation error now only depends on $D$ through the (inverse) gap between the $D$-th and $(D + 1)$-th eigenvalues. Note that using the results of section 2, we would have obtained exactly the same bound for estimating the $D$-th eigenvector only – or even a worse bound since $\widetilde{\delta}_D = \delta_D \wedge \delta_{D-1}$ appears in this case. Thus, at least from the point of view of this technique (which could still yield suboptimal

bounds), there is no increase of complexity between estimating the $D$-th eigenvector alone and estimating the span of the first $D$ eigenvectors.

Note that the inclusion (3) can be interpreted geometrically by saying that for any vector in $\widehat{S}_D$, the tangent of the angle between this vector and its projection on $S_D$ is upper bounded by $B_D/\sqrt{n}$, which we can interpret as a stability property.

**Comment about the Centered Case.**   In the actual (K)PCA procedure, the data is actually first empirically recentered, so that one has to consider the centered covariance operator $\overline{C}$ and its empirical counterpart $\overline{C}_n$. A result similar to Theorem 4 also holds in this case (up to some additional constant factors). Indeed, a result similar to Lemma 1 holds for the recentered operators [2]. Combined again with Theorem 3, this allows to come to similar conclusions for the "true" centered KPCA.

## 4   Conclusion and Discussion

In this paper, finite sample size confidence bounds of the eigenspaces of Kernel-PCA (the $D$-eigenspaces of the empirical covariance operator) are provided using tools of operator perturbation theory. This provides a first step towards an in-depth complexity analysis of algorithms using KPCA as pre-processing, and towards taking into account the randomness of the obtained models (e.g. [3]). We proved a bound in which the complexity factor for estimating the eigenspace $S_D$ by its empirical counterpart depends only on the inverse gap between the $D$-th and $(D+1)$-th eigenvalues. In addition to the previously cited works, we take into account the centering of the data and obtain comparable rates.

In this work we assumed for simplicity of notation the eigenvalues to be simple. In the case the covariance operator $C$ has nonzero eigenvalues with multiplicities $m_1, m_2, \dots$ possibly larger than one, the analysis remains the same except for one point: we have to assume that the dimension $D$ of the subspaces considered is of the form $m_1 + \cdots + m_r$ for a certain $r$. This could seem restrictive in comparison with the results obtained for estimating the sum of the first $D$ eigenvalues themselves [2] (which is linked to the reconstruction error in KPCA) where no such restriction appears. However, it should be clear that we need this restriction when considering $D-$eigenspaces themselves since the target space has to be unequivocally defined, otherwise convergence cannot occur. Thus, it can happen in this special case that the reconstruction error converges while the projection space itself does not. Finally, a common point of the two analyses (over the spectrum and over the eigenspaces) lies in the fact that the bounds involve an inverse gap in the eigenvalues of the true covariance operator.

Finally, how tight are these bounds and do they at least carry some correct qualitative information about the behavior of the eigenspaces? Asymptotic results (central limit Theorems) in [6, 4] always provide the correct goal to shoot for since they actually give the limit distributions of these quantities. They imply that there is still important ground to cover before bridging the gap between asymptotic and non-asymptotic. This of course opens directions for future work.

**Acknowledgements:** This work was supported in part by the PASCAL Network of Excellence (EU # 506778).

## A   Appendix: proofs.

**Proof of Lemma 1.**   This lemma is proved in [9]. We give a short proof for the sake of completness. $\|C_n - C\| = \|\frac{1}{n}\sum_{i=1}^{n} C_{X_i} - \mathbb{E}[C_X]\|$ with $\|C_X\| = \|\varphi(X) \otimes \varphi(X)^*\| = k(X, X) \leq M$. We can apply the bounded difference inequality to the variable $\|C_n - C\|$,

so that with probability greater than $1 - e^{-\xi}$, $\|C_n - C\| \leq \mathbb{E}\left[\|C_n - C\|\right] + 2M\sqrt{\frac{\xi}{2n}}$.

Moreover, by Jensen's inequality $\mathbb{E}\left[\|C_n - C\|\right] \leq \mathbb{E}\left[\|\frac{1}{n}\sum_{i=1}^n C_{X_i} - \mathbb{E}\left[C_X\right]\|^2\right]^{\frac{1}{2}}$, and simple calculations leads to $\mathbb{E}\left[\|\frac{1}{n}\sum_{i=1}^n C_{X_i} - \mathbb{E}\left[C_X\right]\|^2\right] = \frac{1}{n}\mathbb{E}\left[\|C_X - \mathbb{E}\left[C_X\right]\|^2\right] \leq \frac{4M^2}{n}$. This concludes the proof of lemma 1. $\qquad\square$

**Proof of Theorem 3.** The variation of this proof with respect to Theorem 5.2 in [7] is (a) to work directly in a (infinite-dimensional) Hilbert space, requiring extra caution for some details and (b) obtaining an improved bound by considering $D$-eigenspaces at once.

The key property of Hilbert-Schmidt operators allowing to work directly in a infinite dimensional setting is that $HS(\mathcal{H})$ is a both right and left ideal of $\mathcal{L}_c(\mathcal{H}, \mathcal{H})$, the Banach space of all continuous linear operators of $\mathcal{H}$ endowed with the operator norm $\|.\|_{\mathrm{op}}$. Indeed, $\forall T \in HS(\mathcal{H})$, $\forall S \in \mathcal{L}_c(\mathcal{H}, \mathcal{H})$, $TS$ and $ST$ belong to $HS(\mathcal{H})$ with

$$\|TS\| \leq \|T\| \|S\|_{\mathrm{op}} \quad \text{and} \quad \|ST\| \leq \|T\| \|S\|_{\mathrm{op}}. \tag{4}$$

The spectrum of an Hilbert-Schmidt operator $T$ is denoted $\Lambda(T)$ and the sequence of eigenvalues in non-increasing order is denoted $\lambda(T) = (\lambda_1(T) \geq \lambda_2(T) \geq \ldots)$. In the following, $P^D(T)$ denotes the orthogonal projector onto the $D$-eigenspace of $T$.

The Hoffmann-Wielandt inequality in infinite dimensional setting[1] yields that:

$$\|\lambda(A) - \lambda(A + B)\|_{\ell_2} \leq \|B\| \leq \frac{\delta_D}{2}. \tag{5}$$

implying in particular that

$$\forall i > 0, \qquad |\lambda_i(A) - \lambda_i(A + B)| \leq \frac{\delta_D}{2}. \tag{6}$$

Results found in [5] p.39 yield the formula

$$P^D(A) - P^D(A + B) = -\frac{1}{2i\pi}\int_\gamma (R_A(z) - R_{A+B}(z))dz \in \mathcal{L}_c(\mathcal{H}, \mathcal{H}). \tag{7}$$

where $R_A(z) = (A - z\,Id)^{-1}$ is the resolvent of $A$, provided that $\gamma$ is a simple closed curve in $\mathbb{C}$ enclosing exactly the first $D$ eigenvalues of $A$ and $(A + B)$. Moreover, the same reference (p.60) states that for $\xi$ in the complementary of $\Lambda(A)$,

$$\|R_A(\xi)\|_{\mathrm{op}} = dist(\xi, \Lambda(A))^{-1}. \tag{8}$$

The proof of the theorem now relies on the simple choice for the closed curve $\gamma$ in (7), drawn in the picture below and consisting of three straight lines and a semi-circle of radius $L$. For all $L > \frac{\delta_D}{2}$, $\gamma$ intersect neither the eigenspectrum of $A$ (by equation (6)) nor the eigenspectrum of $A + B$. Moreover, the eigenvalues of $A$ (resp. $A + B$) enclosed by $\gamma$ are exactly $\lambda_1(A), \ldots, \lambda_D(A)$ (resp. $\lambda_1(A + B), \ldots, \lambda_D(A + B)$).

Moreover, for $z \in \gamma$, $T(z) = R_A(z) - R_{A+B}(z) = -R_{A+B}(z)BR_A(z)$ belongs to $HS(\mathcal{H})$ and depends continuously on $z$ by (4). Consequently,

$$\|P^D(A) - P^D(A + B)\| \leq \frac{1}{2\pi}\int_a^b \|(R_A - R_{A+B})(\gamma(t))\| \, |\gamma'(t)| dt.$$

Let $S_N = \sum_{n=0}^N (-1)^n (R_A(z)B)^n R_A(z)$. $R_{A+B}(z) = (Id + R_A(z)B)^{-1}R_A(z)$ and, for $z \in \gamma$ and $L > \delta_D$,

$$\|R_A(z)B\|_{\mathrm{op}} \leq \|R_A(z)\|_{\mathrm{op}}\|B\| \leq \frac{\delta_D}{2\,dist(z, \Lambda(A))} \leq \frac{1}{2},$$

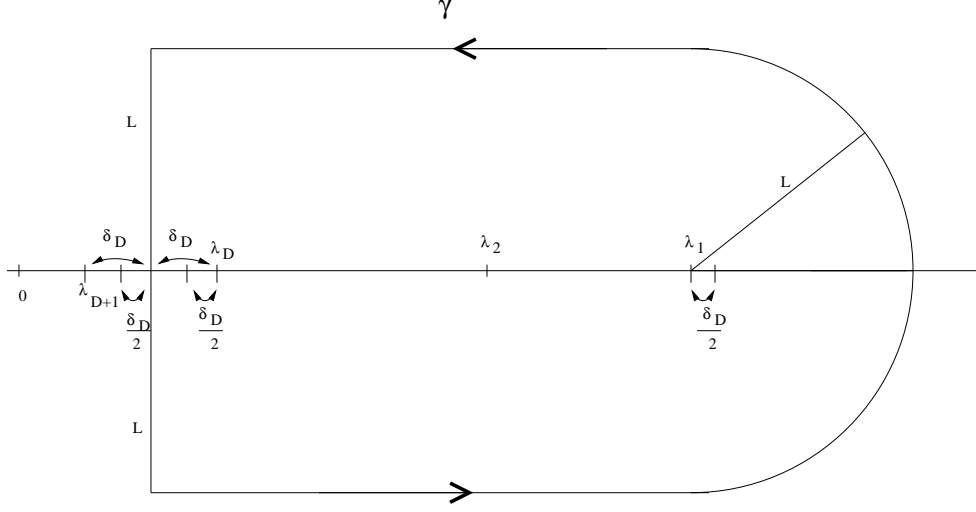

imply that $S_N \xrightarrow{\|\cdot\|_{\text{op}}} R_{A+B}(z)$ (uniformly for $z \in \gamma$). Using property (4), since $B \in HS(\mathcal{H})$, $S_N B R_A(z) \xrightarrow{\|\cdot\|} R_{A+B}(z) B R_A(z) = R_{A+B}(z) - R_A(z)$. Finally,

$$R_A(z) - R_{A+B}(z) = \sum_{n \geq 1} (-1)^n (R_A(z) B)^n R_A(z)$$

where the series converges in $HS(\mathcal{H})$, uniformly in $z \in \gamma$. Using again property (4) and (8) implies

$$\|(R_A - R_{A+B})(\gamma(t))\| \leq \sum_{n \geq 1} \|R_A(\gamma(t))\|_{\text{op}}^{n+1} \|B\|^n \leq \sum_{n \geq 1} \frac{\|B\|^n}{dist^{n+1}(\gamma(t), \Lambda(A))}$$

Finally, since for $L > \delta_D$, $\|B\| \leq \frac{\delta_D}{2} \leq \frac{dist(\gamma(t), \Lambda(A))}{2}$,

$$\|P^D(A) - P^D(A+B)\| \leq \frac{\|B\|}{\pi} \int_a^b \frac{1}{dist^2(\gamma(t), \Lambda(A))} |\gamma'(t)| dt.$$

Splitting the last integral into four parts according to the definition of the contour $\gamma$, we obtain

$$\int_a^b \frac{1}{dist^2(\gamma(t), \Lambda(A))} |\gamma'(t)| dt \leq \frac{2\arctan(\frac{L}{\delta_D})}{\delta_D} + \frac{\pi}{L} + 2\frac{\mu_1(A) - (\mu_D(A) - \delta_D)}{L^2},$$

and letting $L$ goes to infinity leads to the result. $\qquad\square$

**Proof of Theorem 4.** Lemma 1 and Theorem 3 yield inequality (2). Together with assumption $n \geq B_D^2$ it implies $\|P_{S_D} - P_{\widehat{S}_D}\| \leq \frac{1}{2}$. Let $f \in \widehat{S}_D$: $f = P_{S_D}(f) + P_{S_D^\perp}(f)$. Lemma 5 below with $F = S_D$ and $G = \widehat{S}_D$, and the fact that the operator norm is bounded by the Hilbert-Schmidt norm imply that

$$\|P_{S_D^\perp}(f)\|_{\mathcal{H}_k}^2 \leq \frac{4}{3} \|P_{S_D} - P_{\widehat{S}_D}\|^2 \|P_{S_D}(f)\|_{\mathcal{H}_k}^2.$$

Gathering the different inequalities, Theorem 4 is proved. $\qquad\square$

**Lemma 5** *Let $F$ and $G$ be two vector subspaces of $\mathcal{H}$ such that $\|P_F - P_G\|_{\text{op}} \leq \frac{1}{2}$. Then the following bound holds:*

$$\forall f \in G , \ \|P_{F^\perp}(f)\|_{\mathcal{H}}^2 \leq \frac{4}{3} \|P_F - P_G\|_{\text{op}}^2 \|P_F(f)\|_{\mathcal{H}}^2.$$

**Proof of Lemma 5.** For $f \in G$, we have $P_G(f) = f$, hence

$$\|P_{F^\perp}(f)\|^2 = \|f - P_F(f)\|^2 = \|(P_G - P_F)(f)\|^2$$
$$\leq \|P_F - P_G\|_{\text{op}}^2 \|f\|^2$$
$$= \|P_F - P_G\|_{\text{op}}^2 \left(\|P_F(f)\|^2 + \|P_{F^\perp}(f)\|^2\right)$$

gathering the terms containing $\|P_{F^\perp}(f)\|^2$ on the left-hand side and using $\|P_F - P_G\|_{\text{op}}^2 \leq 1/4$ leads to the conclusion. $\qquad\square$

# References

[1] R. Bhatia and L. Elsner. The Hoffman-Wielandt inequality in infinite dimensions. *Proc.Indian Acad.Sci(Math. Sci.) 104 (3), p. 483-494*, 1994.

[2] G. Blanchard, O. Bousquet, and L. Zwald. Statistical Properties of Kernel Principal Component Analysis. *Proceedings of the 17th. Conference on Learning Theory (COLT 2004)*, p. 594–608. Springer, 2004.

[3] G. Blanchard, P. Massart, R. Vert, and L. Zwald. Kernel projection machine: a new tool for pattern recognition. *Proceedings of the 18th. Neural Information Processing System (NIPS 2004)*, p. 1649–1656. MIT Press, 2004.

[4] J. Dauxois, A. Pousse, and Y. Romain. Asymptotic theory for the Principal Component Analysis of a vector random function: some applications to statistical inference. *Journal of multivariate analysis* 12, 136-154, 1982.

[5] T. Kato. *Perturbation Theory for Linear Operators*. New-York: Springer-Verlag, 1966.

[6] V. Koltchinskii. Asymptotics of spectral projections of some random matrices approximating integral operators. *Progress in Probability*, 43:191–227, 1998.

[7] V. Koltchinskii and E. Giné. Random matrix approximation of spectra of integral operators. *Bernoulli*, 6(1):113–167, 2000.

[8] B. Schölkopf, A. J. Smola, and K.-R. Müller. Nonlinear component analysis as a kernel eigenvalue problem. *Neural Computation*, 10:1299–1319, 1998.

[9] J. Shawe-Taylor and N. Cristianini. Estimating the moments of a random vector with applications. *Proceedings of the GRETSI 2003 Conference*, p. 47-52, 2003.

[10] J. Shawe-Taylor, C. Williams, N. Cristianini, and J. Kandola. On the eigenspectrum of the Gram matrix and the generalisation error of Kernel PCA. *IEEE Transactions on Information Theory 51 (7)*, p. 2510-2522, 2005.

[11] U. von Luxburg, M. Belkin, and O. Bousquet. Consistency of spectral clustering. Technical Report 134, Max Planck Institute for Biological Cybernetics, 2004.

[12] U. von Luxburg, O. Bousquet, and M. Belkin. On the convergence of spectral clustering on random samples: the normalized case. *Proceedings of the 17th Annual Conference on Learning Theory (COLT 2004)*, p. 457–471. Springer, 2004.

[13] C. K. I. Williams and M. Seeger. The effect of the input density distribution on kernel-based classifiers. *Proceedings of the 17th International Conference on Machine Learning (ICML)*, p. 1159–1166. Morgan Kaufmann, 2000.
